# EXPERIMENTAL DEMONSTRATIONS OF OPTICAL NEURAL COMPUTERS

Ken Hsu, David Brady, and Demetri Psaltis
Department of Electrical Engineering
California Institute of Technology
Pasadena, CA 91125

## ABSTRACT

We describe two expriments in optical neural computing. In the first a closed optical feedback loop is used to implement auto-associative image recall. In the second a perceptron-like learning algorithm is implemented with photorefractive holography.

## INTRODUCTION

The hardware needs of many neural computing systems are well matched with the capabilities of optical systems[1,2,3]. The high interconnectivity required by neural computers can be simply implemented in optics because channels for optical signals may be superimposed in three dimensions with little or no cross coupling. Since these channels may be formed holographically, optical neural systems can be designed to create and maintain interconnections very simply. Thus the optical system designer can to a large extent avoid the analytical and topological problems of determining individual interconnections for a given neural system and constructing physical paths for these interconnections.

An archetypical design for a single layer of an optical neural computer is shown in Fig. 1. Nonlinear thresholding elements, neurons, are arranged on two dimensional planes which are interconnected via the third dimension by holographic elements. The key concerns in implementing this design involve the need for suitable nonlinearities for the neural planes and high capacity, easily modifiable holographic elements. While it is possible to implement the neural function using entirely optical nonlinearities, for example using etalon arrays[4], optoelectronic two dimensional spatial light modulators (2D SLMs) suitable for this purpose are more readily available. and their properties, i.e. speed and resolution, are well matched with the requirements of neural computation and the limitations imposed on the system by the holographic interconnections[5,6]. Just as the main advantage of optics in connectionist machines is the fact that an optical system is generally linear and thus allows the superposition of connections, the main disadvantage of optics is that good optical nonlinearities are hard to obtain. Thus most SLMs are optoelectronic with a non-linearity mediated by electronic effects. The need for optical nonlinearities arises again when we consider the formation of modifiable optical interconnections, which must be an all optical process. In selecting

a holographic material for a neural computing application we would like to have the capability of real-time recording and slow erasure. Materials such as photographic film can provide this only with an impractical fixing process. Photorefractive crystals are nonlinear optical materials that promise to have a relatively fast recording response and long term memory[4,5,6,7,8].

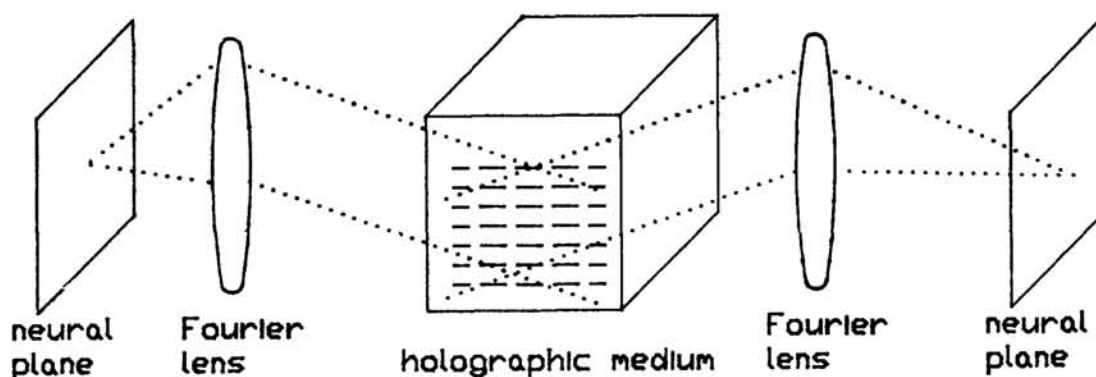

neural plane    Fourier lens    holographic medium    Fourier lens    neural plane

Figure 1. Optical neural computer architecture.

In this paper we describe two experimental implementations of optical neural computers which demonstrate how currently available optical devices may be used in this application. The first experiment we describe involves an optical associative loop which uses feedback through a neural plane in the form of a pinhole array and a separate thresholding plane to implement associate regeneration of stored patterns from correlated inputs. This experiment demonstrates the input-output dynamics of an optical neural computer similar to that shown in Fig. 1, implemented using the Hughes Liquid Crystal Light Valve. The second experiment we describe is a single neuron optical perceptron implemented with a photorefractive crystal. This experiment demonstrates how the learning dynamics of long term memory may be controlled optically. By combining these two experiments we should eventually be able to construct high capacity adaptive optical neural computers.

## OPTICAL ASSOCIATIVE LOOP

A schematic diagram of the optical associative memory loop is shown in Fig. 2. It is comprised of two cascaded Vander Lugt correlators[9]. The input section of the system from the threshold device P1 through the first hologram P2 to the pinhole array P3 forms the first correlator. The feedback section from P3 through the second hologram P4 back to the threshold device P1 forms the second correlator. An array of pinholes sits on the back focal plane of L2, which coincides with the front focal plane of L3. The purpose of the pinholes is to link the first and the second (reversed) correlator to form a closed optical feedback loop[10].

There are two phases in operating this optical loop, the learning phase and the recal phase. In the learning phase, the images to be stored are spatially multiplexed and entered simultaneously on the threshold device. The

thresholded images are Fourier transformed by the lens L1. The Fourier spectrum and a plane wave reference beam interfere at the plane P2 and record a Fourier transform hologram. This hologram is moved to plane P4 as our stored memory. We then reconstruct the images from the memory to form a new input to make a second Fourier transform hologram that will stay at plane P2. This completes the learning phase. In the recalling phase an input is imaged on the threshold device. This image is correlated with the reference images in the hologram at P2. If the correlation between the input and one of the stored images is high a bright peak appears at one of the pinholes. This peak is sampled by the pinhole to reconstruct the stored image from the hologram at P4. The reconstructed beam is then imaged back to the threshold device to form a closed loop. If the overall optical gain in the loop exceeds the loss the loop signal will grow until the threshold device is saturated. In this case, we can cutoff the external input image and the optical loop will be latched at the stable memory.

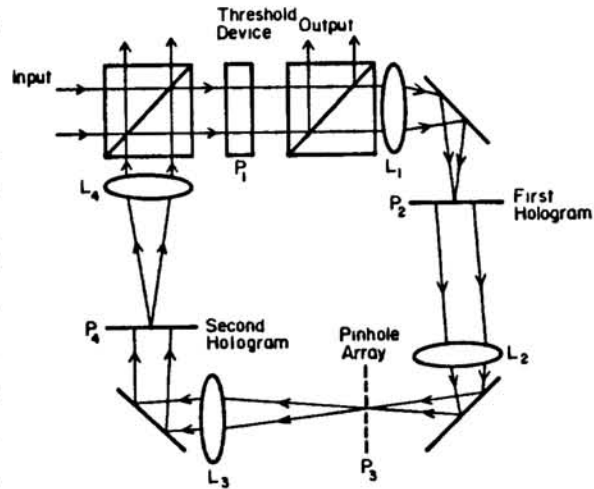

Figure. 2. All-optical associative loop. The threshold device is a LCLV, and the holograms are thermoplastic plates.

The key elements in this optical loop are the holograms, the pinhole array, and the threshold device. If we put a mirror[10] or a phase conjugate mirror[7,11] at the pinhole plane P3 to reflect the correlation signal back through the system then we only need one hologram to form a closed loop. The use of two holograms, however, improves system performance. We make the hologram at P2 with a high pass characteristic so that the input section of the loop has high spectral discrimination. On the other hand we want the images to be reconstructed with high fidelity to the original images. Thus the hologram at plane P4 must have broadband characteristics. We use a diffuser to achieve this when making this hologram. Fig. 3a shows the original images. Fig. 3b and Fig. 3c are the images reconstructed from first and second holograms, respectively. As desired, Fig. 3b is a high pass version of the stored image while Fig. 3c is broadband.

Each of the pinholes at the correlation plane P3 has a diameter of 60 $\mu m$. The separations between the pinholes correspond to the separations of the input images at plane P1. If one of the stored images appears at P1 there will be a bright spot at the corresponding pinhole on plane P3. If the input image shifts to the position of another image the correlation peak will also

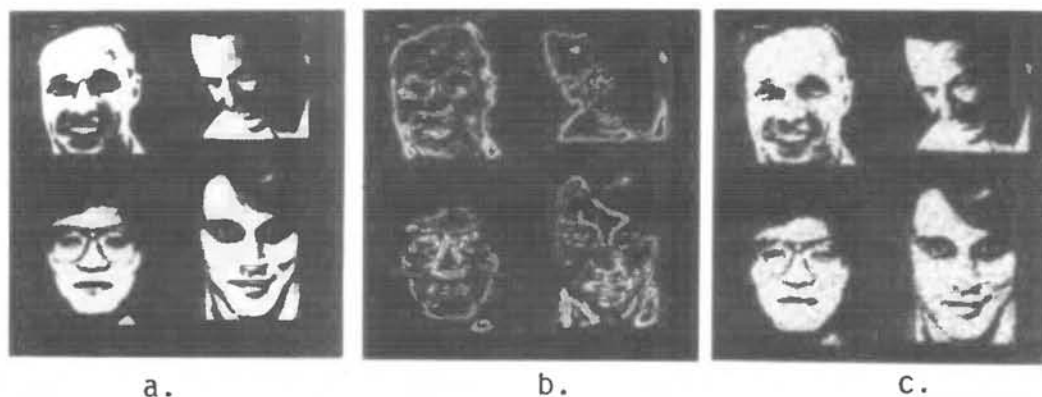

Figure 3. (a) The original images. (b)The reconstructed images from the high-pass hologram **P2**. (c) The reconstructed images from the band-pass hologram **P4**.

shift to another pinhole. But if the shift is not an exact image spacing the correlation peak can not pass the pinhole and we lose the feedback signal. Therefore this is a loop with "discrete" shift invariance. Without the pinholes the cross-correlation noise and the auto-correlation peak will be fed back to the loop together and the reconstructed images won't be recognizable. There is a compromise between the pinhole size and the loop performance. Small pinholes allow good memory discrimination and sharp reconstructed images, but can cut the signal to below the level that can be detected by the threshold device and reduce the tolerance of the system to shifts in the input. The function of the pinhole array in this system might also be met by a nonlinear spatial light modulator, in which case we can achieve full shift invariance[12].

The threshold device at plane P1 is a Hughes Liquid Crystal Light Valve. The device has a resolution of 16 lp/mm and uniform aperture of 1 inch diameter. This gives us about 160,000 neurons at P1. In order to compensate for the optical loss in the loop, which is on the order of $10^{-5}$, we need the neurons to provide gain on the order of $10^5$. In our system this is achieved by placing a Hamamatsu image intensifier at the write side of the LCLV. Since the microchannel plate of the image intensifier can give gains of $10^4$, the combination of the LCLV and the image intensifier can give gains of $10^6$ with sensitivity down to $nW/cm^2$. The optical gain in the loop can be adjusted by changing the gain of the image intensifier.

Since the activity of neurons and the dynamics of the memory loop is a continuously evolving phenomenon, we need to have a real time device to monitor and record this behavior. We do this by using a prism beam splitter to take part of the read out beam from the LCLV and image it onto a CCD camera. The output is displayed on a CRT monitor and also recorded on a video tape recorder. Unfortunately, in a paper we can only show static pictures taken from the screen. We put a window at the CCD plane so that each time we can pick up one of the stored images. Fig. 4a shows the read out image

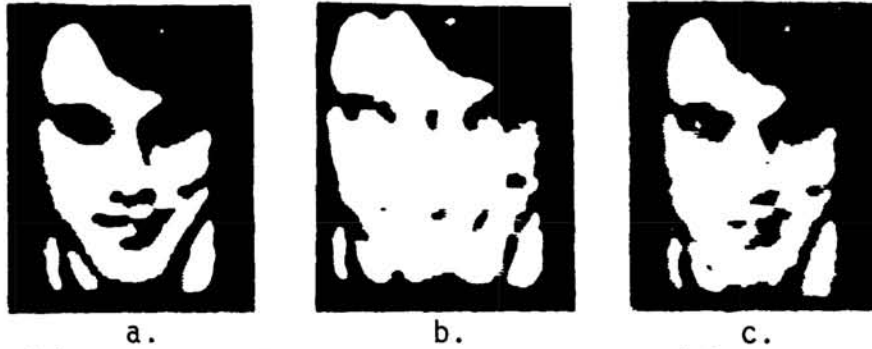

a.          b.          c.

Figure 4. (a) The external input to the optical loop. (b) The feedback image superimposed with the input image. (c) The latched loop image.

from the LCLV which comes from the external input shifted away from its stored position. This shift moves its correlation peak so that it does not match the position of the pinhole. Thus there is no feedback signal going through the loop. If we cut off the input image the read out image will die out with a characteristic time on the order of 50 to 100 ms, corresponding to the response time of the LCLV. Now we shift the input image around trying to search for the correct position. Once the input image comes close enough to the correct position the correlation peak passes through the right pinhole, giving a strong feedback signal superimposed with the external input on the neurons. The total signal then goes through the feedback loop and is amplified continuously until the neurons are saturated. Depending on the optical gain of the neurons the time required for the loop to reach a stable state is between 100 ms and several seconds. Fig. 4b shows the superimposed images of the external input and the loop images. While the feedback signal is shifted somewhat with respect to the input, there is sufficient correlation to induce recall. If the neurons have enough gain then we can cut off the input and the loop stays in its stable state. Otherwise we have to increase the neuron gain until the loop can sustain itself. Fig. 4c shows the image in the loop with the input removed and the memory latched. If we enter another image into the system, again we have to shift the input within the window to search the memory until we are close enough to the correct position. Then the loop will evolve to another stable state and give a correct output.

The input images do not need to match exactly with the memory. Since the neurons can sense and amplify the feedback signal produced by a partial match between the input and a stored image, the stored memory can grow in the loop. Thus the loop has the capability to recall the complete memory from a partial input. Fig. 5a shows the image of a half face input into the system. Fig. 5b shows the overlap of the input with the complete face from the memory. Fig. 5c shows the stable state of the loop after we cut off the external input. In order to have this associative behavior the input must have enough correlation with the stored memory to yield a strong feedback signal. For instance, the loop does not respond to the the presentation of a picture of

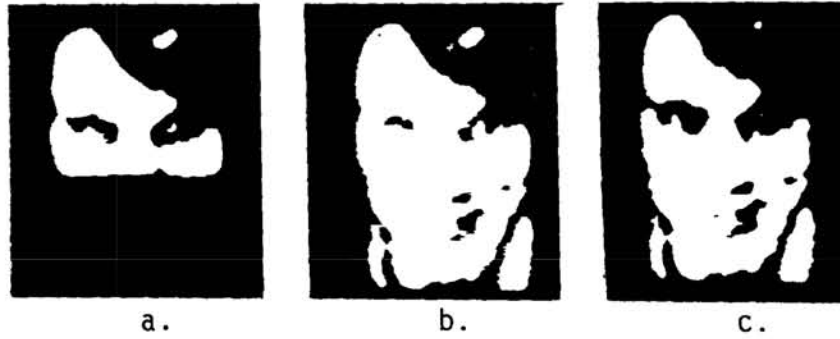

Figure 5. (a) Partial face used as the external input. (b) The superimposed images of the partial input with the complete face recalled by the loop. (c) The complete face latched in the loop.

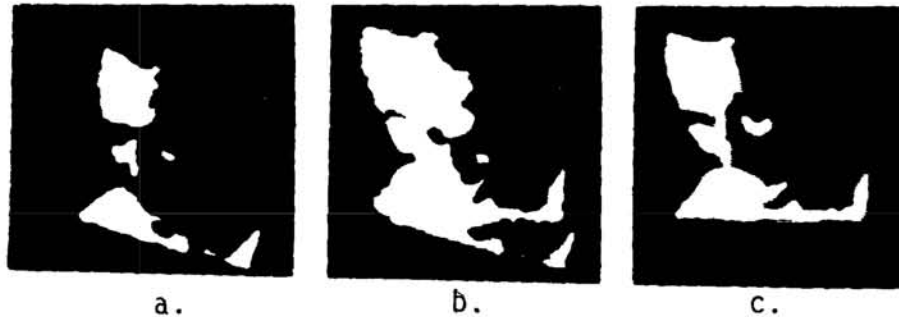

Figure 6. (a) Rotated image used as the external input. (b) The superimposed images of the input with the recalled image from the loop. (c) The image latched in the optical loop.

a person not stored in memory.

Another way to demonstrate the associative behavior of the loop is to use a rotated image as the input. Experiments show that for a small rotation the loop can recognize the image very quickly. As the input is rotated more, it takes longer for the loop to reach a stable state. If it is rotated too much, depending on the neuron gain, the input won't be recognizable. Fig. 6a shows the rotated input. Fig. 6b shows the overlap of loop image with input after we turn on the loop for several seconds. Fig. 6c shows the correct memory recalled from the loop after we cut the input. There is a trade-off between the degree of distortion at the input that the system can tolerate and its ability to discriminate against patterns it has not seen before. In this system the feedback gain (which can be adjusted through the image intensifier) controls this trade-off.

## PHOTOREFRACTIVE PERCEPTRON

Holograms are recorded in photorefractive crystals via the electrooptic modulation of the index of refraction by space charge fields created by the migration of photogenerated charge[13,14]. Photorefractive crystals are attractive for optical neural applications because they may be used to store

long term interactions between a very large number of neurons. While photorefractive recording does not require a development step, the fact that the response is not instantaneous allows the crystal to store long term traces of the learning process. Since the photorefractive effect arises from the reversible redistribution of a fixed pool of charge among a fixed set of optically addressable trapping sites, the photorefractive response of a crystal does not deteriorate with exposure. Finally, the fact that photorefractive holograms may extend over the entire volume of the crystal has previously been shown to imply that as many as $10^{10}$ interconnections may be stored in a single crystal with the independence of each interconnection guaranteed by an appropriate spatial arrangement of the interconnected neurons[6,5].

In this section we consider a rudimentary optical neural system which uses the dynamics of photorefractive crystals to implement perceptron-like learning. The architecture of this system is shown schematically in Fig. 7. The input to the system, $\bar{x}$, corresponds to a two dimensional pattern recorded from a video monitor onto a liquid crystal light valve. The light valve transfers this pattern on a laser beam. This beam is split into two paths which cross in a photorefractive crystal. The light propagating along each path is focused such that an image of the input pattern is formed on the crystal. The images along both paths are of the same size and are superposed on the crystal, which is assumed to be thinner than the depth of focus of the images. The intensity diffracted from one of the two paths onto the other by a hologram stored in the crystal is isolated by a polarizer and spatially integrated by a single output detector. The thresholded output of this detector corresponds to the output of a neuron in a perceptron.

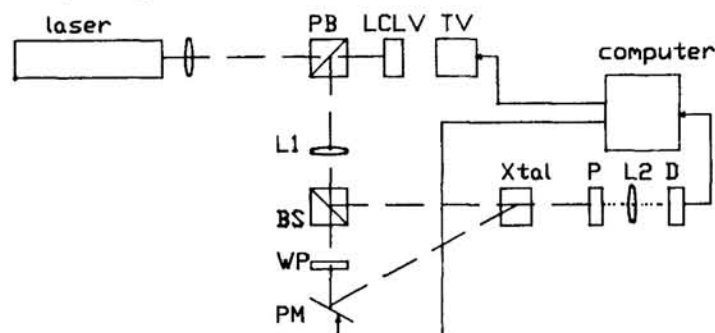

Figure 7. Photorefractive perceptron. PB is a polarizing beam splitter. L1 and L2 are imaging lenses. WP is a quarter waveplate. PM is a piezoelectric mirror. P is a polarizer. D is a detector. Solid lines show electronic control. Dashed lines show the optical path.

The $i^{th}$ component of the input to this system corresponds to the intensity in the $i^{th}$ pixel of the input pattern. The interconnection strength, $w_i$, between the $i^{th}$ input and the output neuron corresponds to the diffraction efficiency of the hologram taking one path into the other at the $i^{th}$ pixel of the image plane. While the dynamics of $w_i$ can be quite complex in some geometries

and crystals, it is possible to show from the band transport model for the photorefractive effect that under certain circumstances the time development of $w_i$ may be modeled by

$$w_i(t) = w_{max} \left| \frac{e^{\frac{-t}{\tau}}}{\tau} \int_0^t m(s) e^{j\phi(s)} e^{\frac{s}{\tau}} ds \right|^2 \tag{1}$$

where $m(s)$ and $\phi(s)$ are the modulation depth and phase, respectively, of the interference pattern formed in the crystal between the light in the two paths[15]. $\tau$ is a characteristic time constant for crystal. $\tau$ is inversely proportional to the intensity incident on the $i^{th}$ pixel of the crystal. Using Eqn. 1 it is possible to make $w_i(t)$ take any value between 0 and $w_{max}$ by properly exposing the $i^{th}$ pixel of the crystal to an appropriate modulation depth and intensity. The modulation depth between two optical beams can be adjusted by a variety of simple mechanisms. In Fig. 7 we choose to control $m(t)$ using a mirror mounted on a piezoelectric crystal. By varying the frequency and the amplitude of oscillations in the piezoelectric crystal we can electronically set both $m(t)$ and $\phi(t)$ over a continuous range without changing the intensity in the optical beams or interrupting readout of the system. With this control over $m(t)$ it is possible via the dynamics described in Eqn. (1) to implement any learning algorithm for which $w_i$ can be limited to the range $(0, w_{max})$.

The architecture of Fig. 7 classifies input patterns into two classes according to the thresholded output of the detector. The goal of a learning algorithm for this system is to correctly classify a set of training patterns. The perceptron learning algorithm involves simply testing each training vector and adding training vectors which yield too low an output to the weight vector and subtracting training vectors which yield too high an output from the weight vector until all training vectors are correctly classified[16]. This training algorithm is described by the equation $\Delta w_i = \alpha x_i$ where alpha is positive (negative) if the output for $\bar{x}$ is too low (high). An optical analog of this method is implemented by testing each training pattern and exposing the crystal with each incorrectly classified pattern. Training vectors that yield a high output when a low output is desired are exposed at zero modulation depth. Training vectors that yield a low output when high output is desired are exposed at a modulation depth of one.

The weight vector for the $k+1^{th}$ iteration when erasure occurs in the $k^{th}$ iteration is given by

$$w_i(k+1) = e^{\frac{-2\Delta t}{\tau}} w_i(k) \approx (1 - \frac{2\Delta t}{\tau}) w_i(k) \tag{2}$$

where we assume that the exposure time, $\Delta t$, is much less than $\tau$. Note that since $\tau$ is inversely proportional to the intensity in the $i^{th}$ pixel, the change in

$w_i$ is proportional to the $i^{th}$ input. The weight vector at the $k + 1^{th}$ iteration when recording occurs in the $k^{th}$ iteration is given by

$$w_i(k+1) = e^{\frac{-2\Delta t}{\tau}} w_i(k) + 2\sqrt{w_i(k)w_{max}} e^{\frac{-\Delta t}{\tau}}(1 - e^{\frac{-\Delta t}{\tau}}) + w_{max}(1 - e^{\frac{-\Delta t}{\tau}})^2 \quad (3)$$

To lowest order in $\frac{\Delta t}{\tau}$ and $\frac{w_i}{w_{max}}$, Eqn. (3) yields

$$w_i(k + 1) = w_i(k) + 2\sqrt{w_i(k)w_{max}}(\frac{\Delta t}{\tau}) + w_{max}(\frac{\Delta t}{\tau})^2 \quad (4)$$

Once again the change in $w_i$ is proportional to the $i^{th}$ input.

We have implemented the architecture of Fig. 7 using a SBN60:Ce crystal provided by the Rockwell International Science Center. We used the 488 nm line of an argon ion laser to record holograms in this crystal. Most of the patterns we considered were laid out on $10 \times 10$ grids of pixels, thus allowing 100 input channels. Ultimately, the number of channels which may be achieved using this architecture is limited by the number of pixels which may be imaged onto the crystal with a depth of focus sufficient to isolate each pixel along the length of the crystal.

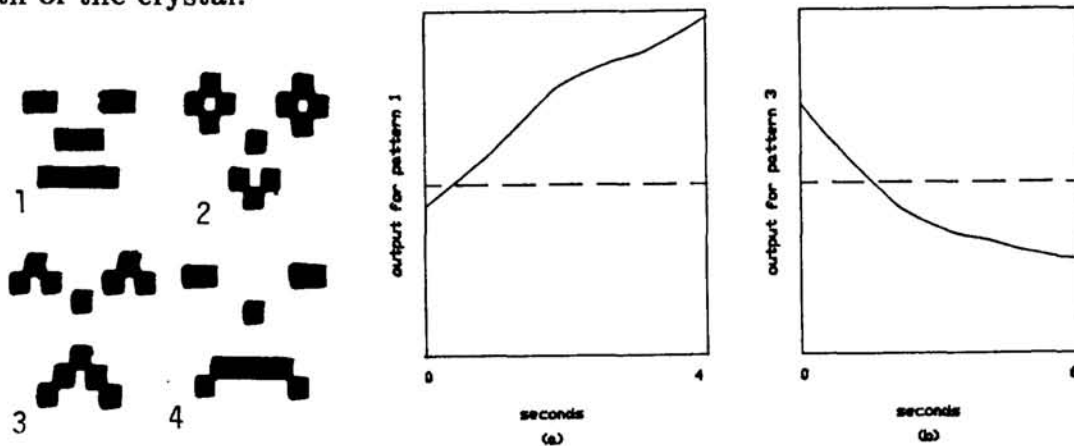

Figure 8. Training patterns.　Figure 9. Output in the second training cycle.

Using the variation on the perceptron learning algorithm described above with a fixed exposure times $\Delta t_r$ and $\Delta t_e$ for recording and erasing, we have been able to correctly classify various sets of input patterns. One particular set which we used is shown in Fig. 8. In one training sequence, we grouped patterns 1 and 2 together with a high output and patterns 3 and 4 together with a low output. After all four patterns had been presented four times, the system gave the correct output for all patterns. The weights stored in the crystal were corrected seven times, four times by recording and three by erasing. Fig. 9a shows the output of the detector as pattern 1 is recorded in the second learning cycle. The dashed line in this figure corresponds to the threshold level. Fig. 9b shows the output of the detector as pattern 3 is erased in the second learning cycle.

## CONCLUSION

The experiments described in this paper demonstrate how neural network architectures can be implemented using currently available optical devices. By combining the recall dynamics of the first system with the learning capability of the second, we can construct sophisticated optical neural computers.

## ACKNOWLEDGEMENTS

The authors thank Ratnakar Neurgaonkar and Rockwell International for supplying the SBN crystal used in our experiments and Hamamatsu Photonics K.K. for assistance with image intesifiers. We also thank Eung Gi Paek and Kelvin Wagner for their contributions to this research.

This research is supported by the Defense Advanced Research Projects Agency, the Army Research Office, and the Air Force Office of Scientific Research.

## REFERENCES

1. Y. S. Abu-Mostafa and D. Psaltis, Scientific American, pp.88-95, March, 1987.

2. D. Psaltis and N. H. Farhat, Opt. Lett., 10,(2), 98(1985).

3. A. D. Fisher, R. C. Fukuda, and J. N. Lee, Proc. SPIE 625, 196(1986).

4. K. Wagner and D. Psaltis, Appl. opt., 26(23), pp.5061-5076(1987).

5. D. Psaltis, D. Brady, and K. Wagner, Applied optics, March 1988.

6. D. Psaltis, J. Yu, X. G. Gu, and H. Lee, Second Topical Meeting on Optical Computing, Incline Village, Nevada, March 16-18,1987.

7. A. Yariv, S.-K. Kwong, and K. Kyuma, SPIE proc. 613-01,(1986).

8. D. Z. Anderson, Proceedings of the International Conference on Neural Networks, San Diego, June 1987.

9. A. B. Vander Lugt, IEEE Trans. Inform. Theory, IT-10(2), pp.139-145(1964).

10. E. G. Paek and D. Psaltis, Opt. Eng., 26(5), pp.428-433(1987).

11. Y. Owechko, G. J. Dunning, E. Marom, and B. H. Soffer, Appl. Opt. 26,(10),1900(1987).

12. D. Psaltis and J. Hong, Opt. Eng. 26,10(1987).

13. N. V. Kuktarev, V. B. Markov, S. G. Odulov, M. S. Soskin, and V. L. Vinetskii, Ferroelectrics, 22,949(1979).

14. J. Feinberg, D. Heiman, A. R. Tanguay, and R. W. Hellwarth, J. Appl. Phys. 51,1297(1980).

15. T. J. Hall, R. Jaura, L. M. Connors, P. D. Foote, Prog. Quan. Electr. 10,77(1985).

16. F. Rosenblatt, Principles of Neurodynamics: Perceptron and the Theory of Brain Mechanisms, Spartan Books, Washington,(1961).